# Learning Multiple Tasks using Shared Hypotheses

**Koby Crammer**
Department of Electrical Enginering
The Technion - Israel Institute of Technology
Haifa, 32000 Israel
koby@ee.technion.ac.il

**Yishay Mansour**
School of Computer Science
Tel Aviv University
Tel - Aviv 69978
mansour@tau.ac.il

## Abstract

In this work we consider a setting where we have a very large number of related tasks with few examples from each individual task. Rather than either learning each task individually (and having a large generalization error) or learning all the tasks together using a single hypothesis (and suffering a potentially large inherent error), we consider learning a small pool of *shared hypotheses*. Each task is then mapped to a single hypothesis in the pool (hard association). We derive VC dimension generalization bounds for our model, based on the number of tasks, shared hypothesis and the VC dimension of the hypotheses class. We conducted experiments with both synthetic problems and sentiment of reviews, which strongly support our approach.

## 1 Introduction

Consider sentiment analysis task for a set of reviews for different products. Each individual product has only very few reviews, which does not enable reliable learning. Furthermore, reviewers may use different amount and level of superlatives to describe the same sentiment level, or feeling different sentiment level yet describing the product with the same text. For example, one may use the sentence "The product is OK" to describe the highest-satisfaction, while another would use "Its a great product, but not amazing" to describe some notion of disappointment. Should one build individual sentiment predictors, one per product, based on small amount of data, or build a single sentiment predictor for all products, based on mixed input with potentially heterogeneous linguistic usage?

One methodology is to cluster individual products to categories, and run the learning algorithm on the aggregated data. While in some cases the aggregation might be simple, in other cases it might be a challenge. (For example, you can cluster restaurants by the cuisine, by the price, by the location, etc.) In addition, the different tasks might be somewhat different on both domain (text used) or predictions (sentiment association with given text), which may raise the dilemma between clustering related tasks or related domain.

In this work we propose an alternative methodology. Rather than clustering the different tasks *before* the learning, perform it as part of the learning task. Specifically, we consider a very large number of tasks, with only few examples from each domain. The goal is to output a pool of few classifiers, and map each task to a single classifier (or a convex combination of them). The idea is that we can control the complexity of the learning process by deciding on the size of pool of shared classifiers. This is a very natural approach, in such a setting.

Our first objective it to study the generalization bounds for such a simple and natural setting. We start by computing an upper and lower bounds on the VC dimension, showing that the VC dimension is at most $O(T \log k + kd \log(Tkd))$, where $T$ is the number of domains, $k$ the number of shared hypothesis and $d$ the VC dimension of the basic hypothesis class. We also show a lower bound of $\max\{kd \ , \ T \min\{d, \log k\}\}$. This shows that the dependency on the number of tasks ($T$) and

the number of shared hypothesis ($k$) is very different, namely, increasing the number of shared hypothesis increases the VC dimension only logarithmically. This will imply that if we have $N$ examples per task, the generalization error is only $\tilde{O}\left(\sqrt{\frac{\log k}{N} + \frac{dk}{TN}}\right)$ compared to $O\left(\sqrt{\frac{d}{N}}\right)$ when learning each task individually. So we have a significant gain when $\log k \ll N$ and $k \ll T$, which is a realistic case. We also derived a K-means like algorithm to learn such classifiers, of both models and association of tasks and models.

Our experimental results support the general theoretical framework introduced. We conduct experiment with both synthetic problems and sentiment prediction, with number of tasks ranging beween $30 - 370$, some contain as high-as $18$ examples in the training set. Our experimental results strongly support the benefits from the approach we propose here, which attains lower test error compared with learning individual models per task, or a single model for all tasks.

**Related Work**

In the recent years there is increasing body of work on domain adaptation and multi-task learning. In domain adaptation we often assume that the tasks to be performed are very similar to each other, yet the data comes from different distributions, and often there is only unlabeled data from the domain (or task) of interest. Mansour et al. [18] develop theory when distribution of the problem of interest (called target) is a convex combination of other distributions for which samples from each is given.

Ben-David et al. [6] focused in classification and developed a distance between distributions and used it to develop new generalization bounds when training and test examples are not coming from the same distributions. Mansour et al. [19] built on that work and developed new distance and theory for adaptation problems with arbitrary loss functions. See also a recent result of Blanchard et al [7].

Another direction of research is to learn few problems simultaneously, yet, unlike in domain adaptation, assuming examples are coming from the same distribution. Obozinski et al. [20] proposed to learn one model per task, yet find a small set of shared features using mixed-norm regularization. Argyriou et al. [4] took a similar approach, yet with added complexity that the feature space can also be rotated before choosing this small shared set. Ando and Zhang [2], and Amit et al. [1], learn by first finding a linear transformation shared by all tasks, and then individual models per task. The first formulation is not convex, while the later is. Evgeniou [13] and Daume [15] proposed to combine two models, one individual per task and the other shared across all tasks, and combine them at test time, while later Evgeniou et al. [12] proposed to learn one model per task, and force all the models to be close to each other. Finally, there exists large body of work on multi-task learning in the Bayesian setting, where a shared prior is used to connect or related the various tasks [5, 22, 16], while other works [17, 21, 9] are using Gaussian process predictors.

The work most similar to our is of Crammer et al. [11, 10] whom developed theory for learning a model with few datasets from various tasks, assuming they are sampled from the same source. They assumed that the relative error (or a bound over it) is known, and proved generalization bound for that task, their bounds proposed to use some of the datasets, but not all, when building a model for the main task. Yet, it was performed before seeing the data and having the strong assumption of the discrepancy between tasks. We do not assume this knowledge and learn few tasks simultaneously.

## 2 Model

There is a set $\mathcal{T}$ of $T$ tasks and with each task $t$ there is an associated distribution $D_t$ over inputs $(\boldsymbol{x}, y)$, where $\boldsymbol{x} \in \mathbb{R}^r$ and $y \in \mathcal{Y}$. We assume binary classification tasks, i.e., $\mathcal{Y} = \{+1, -1\}$. For each task $t \in \mathcal{T}$ has a sample of size $n_t$ denoted by $S_t = \{(\boldsymbol{x}_{t,i}, y_{t,i}, t)\}_{i=1}^{n_t}$ drawn from $D_t$, where $\boldsymbol{x}_{t,i} \in \mathbb{R}^r$ is the $i$-th example in the $t$-th domain and $y_{t,i} \in \mathcal{Y}$ is the corresponding label. (Note that the name of the domain is part of the example, so there is no uncertainty regarding from which domain the example originated from.)

A *k-shared task classifier* is a pair $(H_k, g)$, where $H_k = \{h_1, \ldots, h_k\} \subset \mathcal{H}$ is a set of $k$ hypotheses from a class of functions $\mathcal{H} = \{h : \mathbb{R}^r \to \mathcal{Y}\}$. The function $g$ maps each task $t \in \mathcal{T}$ to the hypotheses pool $H_k$, where the mapping is to a single hypothesis (hard association). We denote by $K = \{1, \ldots, k\}$ the index set for $H_k$.

In the *hard $k$-shared task classifier*, $g$ maps each task $t \in \mathcal{T}$ to one hypothesis in $h_i \in H_k$, i.e., $g : \mathcal{T} \to K$. Classifier $(H_k, g)$, given an example $(\boldsymbol{x}, t)$, first computes the mapping from the domain name $t$ to the hypotheses $h_i$, where $i = g(t)$, and then predicts using the corresponding function $h_i$, i.e., the prediction is $h_{g(t)}(\boldsymbol{x})$. The class of hard $k$-shared task classifiers using hypotheses class $\mathcal{H}$ includes all such $(H_k, g)$ classifiers, i.e., $f_{H_k,g} : \mathbb{R}^r \times \mathcal{T} \to \mathcal{Y}$, where $f_{H_k,g}(\boldsymbol{x}, t) = h_{g(t)}(\boldsymbol{x})$, and the class is $F_{\mathcal{H},k} = \{f_{H_k,g} : |H_k| = k, H_k \subset \mathcal{H}, g : \mathcal{T} \to K\}$.

## 3  Hard $k$-shared Task Classifiers: Generalization Bounds

We envision the following learning process. Given the training sets $S_t$, for $t \in T$, the learner outputs at the end of the training phase both $H_k$ and $g$, where $H_k$ is composed from $k$ hypotheses $h_1, \ldots, h_k \in \mathcal{H}$ and $g : \mathcal{T} \to K$. Naturally, this implies that there is potentially overfitting in both the selection of $H_k$ and the mapping $g$.

Our main goal in this section is to bound the VC dimension of the resulting hypothesis class $F_{\mathcal{H},k}$, assuming the VC dimension of $\mathcal{H}$ is $d$. We show that the VC dimension of $F_{\mathcal{H},k}$ is at most $O(T \log k + kd \log(Tkd))$ and at least $\Omega(T \log k + dk)$.

**Theorem 1.** *For any hypothesis class $\mathcal{H}$ of VC-dimension $d$, the class of hard $k$-shared task classifiers $F_{\mathcal{H},k}$ has VC dimension at most the minimum between $dT$ and $O(T \log k + kd \log(Tkd))$.*

**Proof:** Our main goal is to derive an upper bound on the number of possible labeling using a hard $k$-shared task classifiers $F_{\mathcal{H},k}$. Once we establish this, we can use Sauer lemma to derive an upper bound on the VC dimension [3]. Let $\Phi_d(m) = \sum_{j=0}^{d} \binom{m}{j}$ be an upper bound on the number of labeling over $m$ examples using a hypothesis class of VC dimension $d$. Let $m = \sum_{t \in \mathcal{T}} n_t$ the total sample size.

We consider all mapping $g$ of the $\mathcal{T}$ tasks to $H_k$, there are $k^T$ such mappings. Fix a particular mapping $g$ where hypothesis $h_j$ has tasks $S^j \subset \mathcal{T}$ assigned to it. (At this point $h_j \in \mathcal{H}$ is not fixed yet, we are only fixing $g$ and the tasks that are mapped to the $j$ hypothesis in $H_k$.) There are $m_j = \sum_{t \in S^j} n_t$ examples for the tasks in $S^j$, and therefore at most $\Phi_d(m_j)$ labeling. (Note that the labeling are using any $h \in \mathcal{H}$.) We can upper bound the numbers of labeling *any* hypothesis pool $H_k$ by $\prod_{j=1}^{k} \Phi_d(m_j)$. Since $m = \sum_j m_j$, this bound is maximized when $m_j = m/k$, and this implies that the number of labeling is upper bounded by $k^T(em/dk)^{dk}$.

Now we would like to upper bound the VC dimension of $F_{\mathcal{H},k}$. When $m$ is equal to the VC dimension we have $2^m$ different labeling induced on the $m$ points. Hence, it has to be the case that,

$$2^m \le k^T \left(\frac{em}{dk}\right)^{kd}.$$

We need to find the largest $m$ for which $m \le kd \log(em/dk) + T \log k \le T \log k + kd \log(e/dk) + kd \log m \le T \log k + kd \log m$ for $dk \ge 3$. Note that for $\alpha \ge 2$ and $\beta \ge 1$, we have that if $m < \alpha + \beta \log(m)$ then $m < \alpha + 16\beta \log(\alpha\beta)$. This implies that

$$m \le T \log k + 16kd \log(Tdk \log k) = O\left(T \log k + kd \log(Tkd)\right) ,$$

which derives an upper bound on the number of points that can be shattered, and hence the VC dimension.

To show the upper bound of $dT$, we simply let each domain select a separate hypothesis from $\mathcal{H}$. Since $\mathcal{H}$ has VC dimension $d$, there are at most $d$ examples that can be shattered in each task, for a total of $dT$. ∎

As an immediate corollary we can derive the following generalization bound, using the standard VC dimension generalization bounds [3]. For simplicity we assume that the distribution over the tasks is uniform, we define the true error as $e(f_{H_k,g}) = \Pr_{(x,y,t)}[f_{H_k,g}(x) \ne y]$, and the empirical (or training) error as

$$\hat{e}(f_{H_k,g}) = \frac{\sum_{t=1}^{T} \sum_{i=1}^{n_t} I[f_{H_k,g}(\boldsymbol{x}_{t,i}) \ne y_{t,i}]}{m}, \tag{1}$$

where $m = \sum_t n_t$ is the sample size, and $I(a) = 1$ iff the predicate $a$ is true. We can now state the following corollary, which follows from standard generalization bounds,

**Input parameters:**  $k$ - number of models to use, $N$ - number of iterations, $\eta$ - fraction of data for split
**Initialize:**
- Set random partition $S_t^1 \cup S_t^2 = S_t$ where $S_t^1 \cap S_t^2 = \emptyset$ and $|S_t^1|/|S_t| = \eta$ for $t = 1 \ldots T$
- Set $g(t) = J_t$ where $J_t$ is drawn uniform from $\{1...k\}$

**For**  $i = 1, \ldots, N$

  1. Set $h_j \leftarrow \mathtt{learn}(\cup_{t \in I_j} S_t^1, \mathcal{H})$ where $I_j = \{i \: : \: g(i) = j\}$.
  2. Set $g(t) = \arg\min_{j=1}^{k} \frac{1}{|S_t^2|} \sum_{(\boldsymbol{x},y) \in S_t^2} I[h_j(\boldsymbol{x}) \neq y]$.

**Set**  $h_j \leftarrow \mathtt{learn}(\cup_{t \in I_j} S_t, \mathcal{H})$ where $I_j = \{i \: : \: g(i) = j\}$.

**Output:**   $f_{H_k,g}(\boldsymbol{x})$ where $H_k = \{h_1, \ldots, h_k\}$

---

Figure 1: The SHAMO algorithm for learning shared models.

**Corollary 2.** *Fix $k$. For any hypothesis class $\mathcal{H}$ of VC-dimension $d$, for any hard $k$-shared task classifier $f = (H_k, g)$ we have that with probability $1 - \delta$,*

$$|e(f) - \hat{e}(f)| = O\left(\sqrt{\frac{(T\log k + kd\log(Tkd))\log(m/T) + \log 1/\delta}{m}}\right).$$

The previous corollary holds for some fixed $k$ known before observing the training data, we now state a bound where $k$ is chosen after seeing the data, together with $g$ and $H_k$. The proof follows from the previous corollary and performing a union bound on the different values of $k$,

**Corollary 3.** *For any hypothesis class $\mathcal{H}$ of VC-dimension $d$, for any $k$, for any hard $k$-shared task classifier $f = (H_k, g)$ we have that with probability $1 - \delta$,*

$$|e(f) - \hat{e}(f)| = O\left(\sqrt{\frac{(T\log k + kd\log(Tkd))\log(m/T) + \log(k/\delta)}{m}}\right).$$

The last two bounds state that empirical error is close to true error under two conditions, first that $T\log k$ is small in compared with $m = \sum_t n_t$. That is, the average number of examples (per task), should be large compared to the log-number-of models. Thus, even with a dozen models, only few tens of examples are suffice. Second, that $kd$ is small compared with $m$. The main point is that if the VC dimension is large and the average number of examples $m/T$ is low, it is possible to compensate if the number of models $k$ is small relative to the number of tasks $T$. Hence, we expect to improve performance over individual models if there are many-tasks, yet we predict with relative few models.

We now show that our upper bound on the VC dimension is almost tight.

**Theorem 4.** *There is a hypothesis class $\mathcal{H}$ of VC-dimension $d$, such that the class of hard $k$-shared task $F_{\mathcal{H},k}$ has VC dimension at least $\max\{kd \: , \: T\min\{d, \log k\}\}$.*

**Proof:** To show the lower bound of $kd$ consider $d$ points that $\mathcal{H}$ shatters, $\boldsymbol{x}_1, \ldots, \boldsymbol{x}_d$. Consider the set of examples $S = \{(\boldsymbol{x}_i, j) : 1 \leq j \leq k, 1 \leq i \leq d\}$. For any labeling of $S$, we can select for each domain $j$ a different hypothesis from $\mathcal{H}$ that agrees with the labeling. Since we have only $k$ different $j$s, we can do it with $k$ functions. Therefore we shatter $S$ and have a lower bound on $kd$.

Let $\ell = \min\{d, \log k\}$, hence the second bound is $T\ell$. Since class $\mathcal{H}$ is of VC dimension $d$, this implies that there are points $x_1, \ldots, x_\ell$ and function $h_1, \ldots h_k \in \mathcal{H}$, such that for any labeling of $x_i$'s there is a hypothesis $h_j$ which is consistent with it. (Since $k$ hypotheses can shatter at most $\log k$ points, we get the dependency on $\log k$.) Let the sample be $S = \{(x_i, t) : 1 \leq i \leq \ell, t \in \mathcal{T}\}$. For any labeling of $S$, when we consider domain $t \in \mathcal{T}$, there is a function in $h_i \in H_k$ which is consistent with the labeling. Therefore the VC dimension is at least $T\ell$. ∎

## 4   Learning with SHAred MOdels (SHAMO) Algorithm

The generalization bound states that we should find a pair $(H_k, g)$ that perform well on the training data and that $k$ would be small a-priori. We assume that there is a learning algorithm from $\mathcal{H}$ called

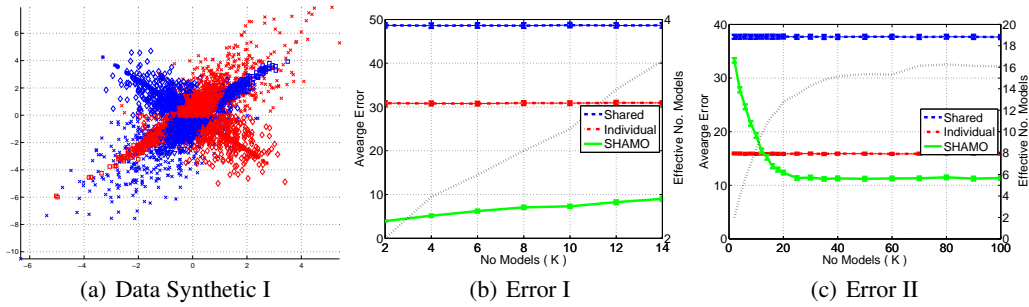

| (a) Data Synthetic I | (b) Error I | (c) Error II |

Figure 2: Left: Illustration of data used in the first experiment. The middle (experiment I) and right (experiment II) panels shows the average error vs k for the three algorithms, and the "effective" number of models vs k (right axis).

with a training set $S$. Formally, we assume that the hypothesis returned by $\hat{h} \leftarrow \texttt{learn}(\texttt{S}, \mathcal{H})$ has lowest training error, that is the algorithm performs empirical risk minimization. We propose to perform an iterative procedure, between two stages, which intuitively is similar to K-means.

In the first stage, the algorithm fixes the assignment function $g$ and find the best $k$ functions $H_k$. This can be performed easily by calling $k$ times any algorithm that learns with the hypothesis class $\mathcal{H}$. On each call the union of the training sets that are assigned by $g$ the same value is fed into the algorithm. Formally, for all $j = 1 \dots k$ set, $h_j \leftarrow \texttt{learn}(\cup_{t \in I_j} S_t, \mathcal{H})$ where $I_j = \{i \; : \; g(i) = j\}$. In the second stage we learn the association $g$ given $H_k$. Here we simply set $g(t)$ to be the model which attains the lowest error evaluated on the training set, that is, $g(t) = \arg\min_{j=1}^{k} \frac{1}{n_t} \sum_{i=1}^{n_t} I[h_j(\boldsymbol{x}_{t,i}) \neq y_{t,i}]$.

This procedure can be repeated for a fixed number of iterations, or until a convergence criteria is met. Specifically, in the experiments below our algorithm iterated between the step exactly 10 times. Clearly, each stage reduces the training error of (1), but how far the resulting hypotheses from the optimal one is not clear.

In the description above the training sets $S_t$ was used twice, once for finding $H_k$ and once for finding $g$. We found in practice that this leads to over-fitting, that is, in the second stage sub-optimal hypotheses are assigned to $g$ if evaluated on the test set (which clearly is not known during training time.) We thus modify the algorithm above, and use only part of the training set for each of the tasks, where these parts not over overlapping. Formally, before performing the iterations the algorithm partitions the training set, into two parts, $S_t^1 \cup S_t^2 = S_t$ where $S_t^1 \cap S_t^2 = \emptyset$. Then the first stage is performed by calling the learning procedure with the first set and the second with the second set. Only after iterations are concluded, we use the entire training set to build models, with out modifying the association function $g$. We call this algorithm SHAMO for learning with shared models. The algorithm is summarized in the Fig. 1.

## 5  Empirical Study

We evaluated our algorithm on both synthetic and real-world sentiment classification task. Training was performed using the averaged-Perceptron [14] executed for 10 iterations. Three methods are evaluated, learning one model per task, called *Individual* below, learning one model for all tasks called *Shared* below, and learning $k$ models using our algorithm, *SHAMO*. We also implemented an online version of a batch algorithm for this setting [4]. *SHAMO* was outperformed it in the majority of experiments. Full details will be included in a long version of this paper.

**Synthetic Data:**  We first report results using synthetic data. We generated 20 dimensional inputs $\boldsymbol{x} \in \mathbb{R}^{20}$. All features were drawn from Gaussian with mean zero. The first two inputs of tasks $t$ were drawn with a covariance specific for that tasks. The remaining 18 features were with isotropic covariance. The label of input $\boldsymbol{x} = (x_1, x_2, ..., x_{20})$ was set to be $\text{sign}(x_2 \cdot s_t)$ where $s_t \in \{-1, +1\}$ with probability half. We generated $T = 200$ such tasks each with 6 training examples (with at least one example from each class), and ran our algorithm for various values of $k$. Models were evaluated on tests sets of size $n = 1,000$ for each task. The results below are averages

over 50 random repetitions of the data generation process. Plot of test set (with $T = 9$ for ease of presentation) appear in the left-panel of Fig. 2, clearly two models are enough to classify all tasks correctly (depending on the value of $s_t$ above), and furthermore, applying the wrong model yields test error of $100\%$. All 6 examples were used both to build models and associating models to tasks.

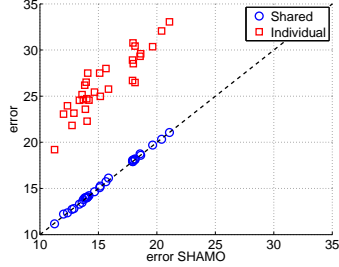

(a) Error of Individual and Shared vs. Error of Shamo

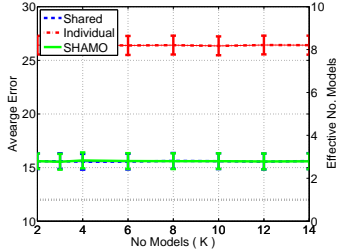

(b) Error vs. k

Figure 3: Results for Data A (31 Tasks, 1 Thresh)

The results are summarized in middle panel of Fig. 2, in which we plot mean error of the three algorithms vs the number of models $k$, with error bars for $95\%$ confidence interval. Since both *Individual* and *Shared* are independent of $k$, the line is flat for them. It is clear that *Shared* performs worst with an average error of $50\%$ (highest line), which is explained by the fact that the test error of half of the models over the other half of the data-sets is about $100\%$. *Individual* performs second, with test error of about $30\%$ obtained by only 6 training examples. Our algorithm, *SHAMO*, performs the best with error of about $5\%$ when allowing $k = 2$ models, and about $10\%$ when allowing $k = 14$ models. The dotted-black line indicates the number of "effective" models per value of $k$, which is the smallest number of models which at least 90 tasks are associated with (exactly) one of them. The corresponding scale is the right axis. Indeed as the number of *possible* models $k$ is increased to $14$, the number of effective models is also increased, but only moderately, from an average of 2 to an average of $3.5$. In other words, only small number of models are used in practice, which avoids severe overfitting.

The next synthetic experiment was performed with 10 target models and more noise. Here, we generated 40 dimensional inputs $x \in \mathbb{R}^{40}$. All features were drawn from Gaussian with mean zero. The first ten inputs of tasks $t$ were drawn with a covariance specific for that tasks. The remaining 30 features were with isotropic covariance. The label of input $x = (x_1, x_2, ...x_{40})$ was set to be $\text{sign}(u_t \cdot (x_1 \ldots x_{10}))$ where $u_t \in \mathbb{R}^{10}$ are a set of 10 orthogonal vectors, chosen uniformly in random. As in the first experiment, we generated $T = 200$ such tasks, each with 25 training examples, and ran *SHAMO* with values of $k$ ranging between 2 and 100. Models were evaluated on tests sets of size $n = 1,000$ for each task. The results below are averages over 50 random repetitions of the data generation process. In these experiments ten models are enough to classify all tasks correctly, yet in this experiment, applying the wrong model yields test error of only $50\%$. Out of the 25 examples available for each task, 7 were used to build models, and the remaining 18 were used to associate models to tasks ($\eta = 7/25$). Lower values cause overfitting, while higher values yield poor models.

The results are summarized in right panel of Fig. 2, in which we plot mean error of the three algorithms vs the number of models $k$, with error bars for $95\%$ confidence interval. The bottom line is similar to the previous experiment. As before, *Shared* performs worst, *Individual* performs second, with test error of about performing second with about $15\%$ obtained with 25 training examples. Our algorithm, *SHAMO*, performs the best with error of about $11\%$ when allowing $k = 22$ models, twice the number of real models. Additionally, it seems that the algorithm was not-overfitting, even when the number of allowed models was set to 100 the performance was the same as setting $k = 25$. One possible explanation is that the algorithm is not using all allowed models, indeed the number of "effective" models (which are associated to $90\%$ of the tasks) grows moderately for number of models greater than 25 (from 14 to 16). In other words, if we allow the algorithm to remove about $10\%$ of the tasks, then only $14 - 16$ models are enough to have about $11\%$ test error on average. It is not clear to us yet, why over-fitting occurred in the first experiment but not in the second.

**Sentiment Data:** We followed Blitzer et.al [8] and evaluated our algorithm also on product reviews from Amazon. We downloaded $2,000$ reviews from 31 categories, such as books, dvd and so on; a total of $62,000$ reviews all together. All reviews were represented using bag-of-unigrams/bigrams, using only features that appeared at least 5 times in all training sets, yielding a dictionary of size $28,775$. The reviews we used were originally labeled with $1, 2, 4, 5$ stars, as reviews with 3 stars were very hard to predict, even with very large amount of data.

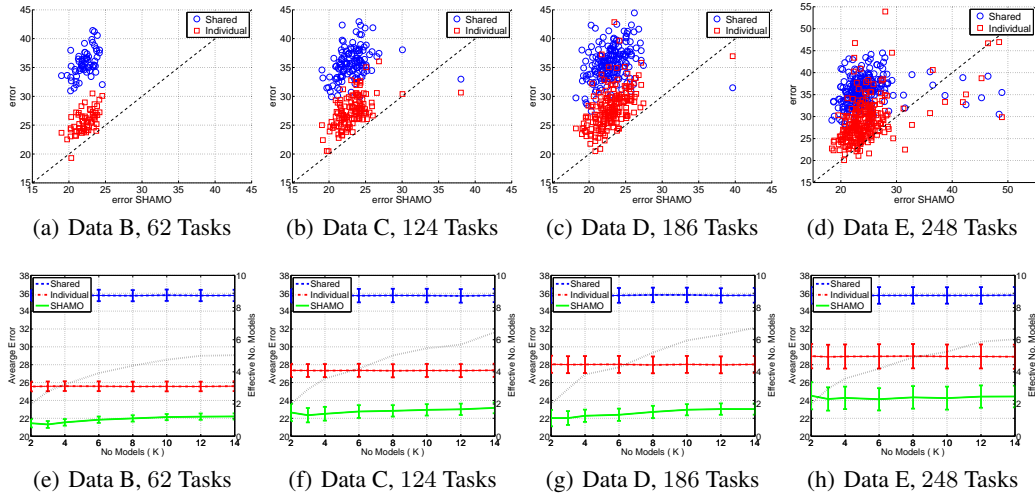

**Figure 4:** Top: test error of *Individual* and *Shared* algorithms vs test error of SHAMO for $k = 14$, for all datasets with 2 thresholds. Bottom: average error vs k for the three algorithms, and the "effective" number of models vs k (right axis).

| Data | Thresholds | No. Tasks | Training Size | Test Size |
|------|-----------|-----------|---------------|-----------|
| A | 1 | 31 | 220 | 1,780 |
| B | 2 | 62 | 108 | 892 |
| C | 2 | 124 | 54 | 446 |
| D | 2 | 186 | 36 | 297 |
| E | 2 | 248 | 27 | 223 |
| F | 3 | 93 | 72 | 592 |
| G | 3 | 186 | 36 | 296 |
| H | 3 | 279 | 24 | 197 |
| I | 3 | 372 | 18 | 148 |

Table 1: Summary of sentiment datasets used.

We generated three binary prediction datasets as follows. In the first dataset, the goal was to predict whether the number of stars associated with a review is above or below 3. Since we focus in the case of many tasks with small amount of data each, we used about $1/9$ of the data for training and the remaining for evaluation. Each set (training and test) contains equal amount of reviews with the $1, 2, 4, 5$ stars. The outcome of this process are 31 tasks, each with 220 training examples and $1,780$ test examples. This dataset is in row *A* of Table 1.

For the second dataset we partitioned all reviews from each category into two equal sets. The prediction problem for the first was to predict if the number of stars is $5$ stars or not (that is, below $5$). For the second set of problems the goal was to predict if the number of stars is $1$ or not. The outcome are 62 tasks with 108 training examples and 892 test examples. We refer to this problem as having 2 thresholds (5 and 1). This dataset is row *B* of Table 1. For the third dataset we partitioned the reviews into three sets, using one of the three goals above - is the number of starts above or below 1, is it above or below 3, and is it above or below 5 - ending up with 93 tasks with 72 training examples and 592 test examples in each. We refer to this problem as having 3 thresholds (5, 3 and 1). This dataset is row *F* in Table 1. Finally, we took each of the last two problems and divided each task into 2, 3 or 4 - rows *C,D,E* (2 thresholds) and, rows *G,H,I* (3 thresholds). Our setting with few thresholds represent different language usages, from mild to strong, for the same level of sentiment.

Unlike in the synthetic experiments training data was either used for building models, or associating models to tasks. That is, we set $|S_t^1| = |S_t^2| = 0.5|S_t|$ for $\eta = 0.5$, and used one half of the examples to build models (set the weights of prediction functions), and the remaining half to evaluate each of the $k$ models on the $T$ tasks, and associating models to tasks. Only after this process ended, we fixed this association and learned models using all training points to build final models.

The results for dataset *A* of single threshold appear in Fig. 3. The top panel shows the error of *Individual* and *Shared* vs *SHAMO* for $k = 14$. Points above the line $y = x$ indicate the superiority of *SHAMO*. Although we used reviews from 31 domains, there is essentially a single task, and thus it is best to combine the data. Indeed, all the red-squares (corresponding to *Individual*) are above the blue-circles (corresponding to *Shared*), indicating that the shared model outperforms individual models. Additionally, all points corresponding to *Shared* lies on the diagonal, indicating that

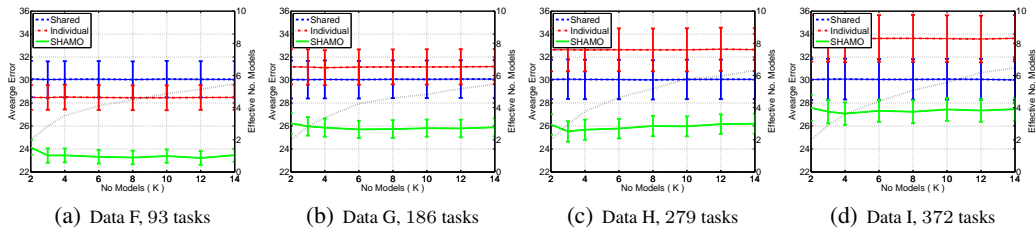

| (a) Data F, 93 tasks | (b) Data G, 186 tasks | (c) Data H, 279 tasks | (d) Data I, 372 tasks |

Figure 5: Average error vs k for the three algorithms, and the "effective" number of models vs k (right axis).

*SHAMO* is performing as well as *Shared*, with error $\sim 16\%$. The bottom panel shows the performance of *SHAMO* vs. $k$. As shown, the error is fixed and is not affected by $k$. This is explained by the black-dashed-line that, as before, shows the number of "effective" models, which is $1$. Even though the algorithm may choose up to $14$ models, it is always using effectively one.

The results for datasets *B-E* all with two thresholds are summarized in Fig. 4. The top panels show the test error of *Individual* and *Shared* algorithms vs test error of *SHAMO* for $k = 14$, with number of tasks increasing from left to right. First, as opposed to dataset *A* with single threshold, in all cases the results for *Shared* are worse than these of *Individual*. This gap is getting smaller with the number of tasks (the clouds are overlapping as we go from left panel to right). intuitively, *Shared* introduces (label) bias as the two thresholds are being treated as one, while *Individual* introduces variance as smaller and smaller training sets are used, as we go from the left panel to the right one, the gap between bias and variance shrinks as the variance is increased. *SHAMO* performs the best as in all plots almost all the points (less in the right plot) are above the line $y = x$. Additionally, the spread of the cloud in the top-panels is getting larger, indicating larger deviation in the performance across different tasks.

The bottom panels of Fig. 4 shows the average test error vs $k$. As *Shared* is not affected by $k$ nor $T$ (as total training examples remains the same), its test error of $36\%$ is fixed across panels. As we have more tasks, and less training examples per task, the test error of *Individual* increases from $25.6\%$ to $28.9\%$ (gap of $3.3\%$). *SHAMO* performs the best, and is also affected from smaller dataset, with test error ranging between $21.8$ and $24.3$, having a smaller gap than *Individual* of $2.5\%$). In all four dataset the optimal number of models is $k = 3$, and there is minor overfitting when using larger values (at most $1\%$). As before the *effective* number of models grows weakly with $k$.

The results for datasets *F-I* all with three thresholds are summarized in Fig. 5, the general trend remains the same, and we highlight only the main differences. First, the gap between *Individual* and *Shared* is much smaller, in some tasks one is better, and in other tasks the other is better. Additionally, for the smallest number of tasks (left) *Individual* is better with a gap of $\sim 1.5\%$, while for largest number of tasks *Individual* is worse with a gap ranging between $1 - 4\%$. This is exactly where the effect of variance of small datasets became stronger than the bias emerging from sharing. Second, in general these dataset are more heterogeneous, as indicated by the larger standard-deviation (longer error-bars than in Fig. 4). As before *SHAMO* performs the best, with optimal performance when $k = 3 - 4$ and is almost not overfitting for larger values of $k$, as the "effective" number of models grows slowly with $k$.

**Summary**

We described theoretical framework for multitask learning using small number of shared models. Our theory suggests that many-tasks can be used to compensate for small number of training examples per task, if one can partition that tasks to few sets, with similar labeling function per set. We also derived a K-means-like algorithm to learn such classifiers of both models and association of taks and models. Our experimental results on both hand-crafted problems and real-world sentiment classification problem strongly support the benefits from the approach, even with very few examples per task. We plan to extend our theory to direct of the optimal splitting of the training data by the algorithm, analyze its convergence properties and perform extensive experiments. We also plan to derive theory and algorithms for soft association of tasks to classifiers.

**Acknowledgements:** The research is partially supported by a grants from ISF, BSF and European Union grant IRG-256479.

# References

[1] Yonatan Amit, Michael Fink, Nathan Srebro, and Shimon Ullman. Uncovering shared structures in multiclass classification. In *ICML*, pages 17–24, 2007.

[2] Rie Kubota Ando and Tong Zhang. A framework for learning predictive structures from multiple tasks and unlabeled data. *Journal of Machine Learning Research*, 6:1817–1853, 2005.

[3] Martin Anthony and Peter L. Bartlett. *Neural Network Learning: Theoretical Foundations*. Cambridge University Press, 1999.

[4] Andreas Argyriou, Theodoros Evgeniou, and Massimiliano Pontil. Convex multi-task feature learning. *Machine Learning*, 73(3):243–272, 2008.

[5] Bart Bakker and Tom Heskes. Task clustering and gating for bayesian multitask learning. *Journal of Machine Learning Research*, 4:83–99, 2003.

[6] Shai Ben-David, John Blitzer, Koby Crammer, Alex Kulesza, Fernando Pereira, and Jennifer Wortman Vaughan. A theory of learning from different domains. *Machine Learning*, 79(1-2):151–175, 2010.

[7] Gilles Blanchard, Gyemin Lee, and Clay Scott. Generalizing from several related classification tasks to a new unlabeled sample. In *NIPS*, 2011.

[8] John Blitzer, Mark Dredze, and Fernando Pereira. Biographies, bollywood, boom-boxes and blenders: Domain adaptation for sentiment classification. In *Association for Computational Linguistics (ACL)*, 2007.

[9] Edwin V. Bonilla, Felix V. Agakov, and Christopher K. I. Williams. Kernel multi-task learning using task-specific features. *Journal of Machine Learning Research - Proceedings Track*, 2:43–50, 2007.

[10] Koby Crammer, Michael Kearns, and Jennifer Wortman. Learning from multiple sources. *Journal of Machine Learning Research*, 9:1757–1774, 2008.

[11] Koby Crammer, Michael J. Kearns, and Jennifer Wortman. Learning from data of variable quality. In *NIPS*, 2005.

[12] Theodoros Evgeniou, Charles A. Micchelli, and Massimiliano Pontil. Learning multiple tasks with kernel methods. *Journal of Machine Learning Research*, 6:615–637, 2005.

[13] Theodoros Evgeniou and Massimiliano Pontil. Regularized multi–task learning. In *KDD*, pages 109–117, 2004.

[14] Y. Freund and R. E. Schapire. Large margin classification using the perceptron algorithm. In *Proceedings of the Eleventh Annual Conference on Computational Learning Theory*, 1998. To appear, *Machine Learning*.

[15] Hal Daumé III. Frustratingly easy domain adaptation. In *ACL*, 2007.

[16] Hal Daumé III. Bayesian multitask learning with latent hierarchies. In *UAI*, pages 135–142, 2009.

[17] Neil D. Lawrence and John C. Platt. Learning to learn with the informative vector machine. In *ICML*, 2004.

[18] Yishay Mansour, Mehryar Mohri, and Afshin Rostamizadeh. Domain adaptation with multiple sources. In *NIPS*, pages 1041–1048, 2008.

[19] Yishay Mansour, Mehryar Mohri, and Afshin Rostamizadeh. Domain adaptation: Learning bounds and algorithms. In *COLT*, 2009.

[20] Guillaume Obozinski, Ben Taskar, and Michael I. Jordan. Joint covariate selection and joint subspace selection for multiple classification problems. *Statistics and Computing*, 20(2):231–252, 2010.

[21] Kai Yu, Volker Tresp, and Anton Schwaighofer. Learning gaussian processes from multiple tasks. In *ICML*, pages 1012–1019, 2005.

[22] Shipeng Yu, Volker Tresp, and Kai Yu. Robust multi-task learning with *t*-processes. In *ICML*, pages 1103–1110, 2007.

